# Unsupervised Template Learning for Fine-Grained Object Recognition

**Shulin Yang**
University of Washington, Seattle, WA 98195
yang@cs.washington.edu

**Liefeng Bo**
ISTC-PC Intel labs, Seattle, WA 98195
liefeng.bo@intel.com

**Jue Wang**
Adobe ATL Labs, Seattle, WA 98103
juewang@adobe.com

**Linda Shapiro**
University of Washington, Seattle, WA 98195
shapiro@cs.washington.edu

## Abstract

Fine-grained recognition refers to a subordinate level of recognition, such as recognizing different species of animals and plants. It differs from recognition of basic categories, such as humans, tables, and computers, in that there are global similarities in shape and structure shared cross different categories, and the differences are in the details of object parts. We suggest that the key to identifying the fine-grained differences lies in finding the right alignment of image regions that contain the same object parts. We propose a template model for the purpose, which captures common shape patterns of object parts, as well as the co-occurrence relation of the shape patterns. Once the image regions are aligned, extracted features are used for classification. Learning of the template model is efficient, and the recognition results we achieve significantly outperform the state-of-the-art algorithms.

## 1 Introduction

Object recognition is a major focus of research in computer vision and machine learning. In the last decade, most of the existing work has been focused on basic recognition tasks: distinguishing different categories of objects, such as table, computer and human. Recently, there is an increasing trend to work on subordinate-level or fine-grained recognition that categorizes similar objects, such as different types of birds or dogs, into their subcategories. The subordinate-level recognition problem differs from the basic-level tasks in that the object differences are more subtle. Fine-grained recognition is generally more difficult than basic-level recognition for both humans and computers, but it will be widely useful if successful in applications such as fisheries (fish recognition), agriculture (farm animal recognition), health care (food recognition), and others.

Cognitive research study has suggested that basic-level recognition is based on comparing the shape of the objects and their parts, whereas subordinate-level recognition is based on comparing appearance details of certain object parts [1]. This suggests that finding the right correspondence of object parts is of great help in recognizing fine-grained differences. For basic-level recognition tasks, spatial pyramid matching [2] is a popular choice that aligns object parts by partitioning the whole image into multiple-level spatial cells. However, spatial pyramid matching may not be the best choice for fine-grained object recognition, since falsely aligned object parts can lead to inaccurate comparisons, as shown in Figure 1.

This work is intended to alleviate the limitations of spatial pyramid matching. Our key observation is that in a fine-grained task, different object categories share commonality in their shape or structure, and the alignment of object parts can be greatly improved by discovering such common

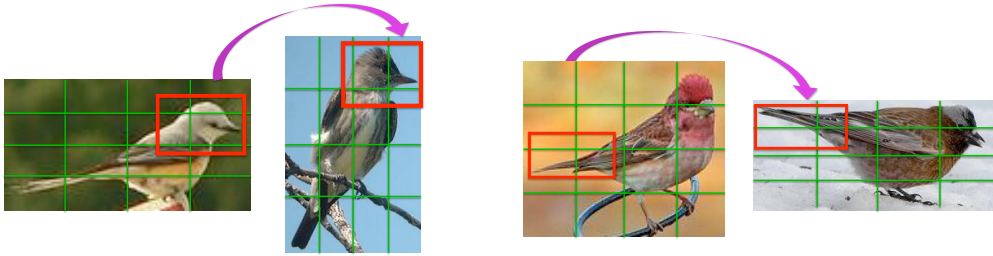

Figure 1: Region alignment by spatial pyramid matching and our approach. Spatial pyramid matching partitions the whole image into regions, without considering visual appearance. A $4 \times 4$ partition leads to misalignment of parts of the birds while a coarse partition (i.e. $2 \times 2$) includes irrelevant features. Our approach aims to align the image regions containing the same object parts (red squares).

shape patterns. For example, bird images from different species may have similar shape patterns in their beaks, tails, feet or bodies. The commonality usually is a part of the global shape, and can be observed in bird images across different species and in different poses. This motivates us to decompose a fine-grained object recognition problem into two sub-problems: 1) aligning image regions that contain the same object part and 2) extracting image features within the aligned image regions. To this end, we propose a template model to align object parts. In our model, a template represents a shape pattern, and the relationship between two shape patterns is captured by the relationship between templates, which reflects the probability of their co-occurrence in the same image. This model is learned using an alternative algorithm, which iterates between detecting aligned image regions, and updating the template model. Kernel descriptor features [3, 4] are then extracted from image regions aligned by the learned templates.

Our model is evaluated on two benchmark datasets: the Caltech-UCSD Bird200 and the Stanford Dogs. Our experimental results suggest that the proposed template model is capable of detecting image regions that correspond to meaningful object parts, and our template-based algorithm outperforms the state-of-the-art fine-grained object recognition algorithms in terms of accuracy.

## 2 Related Work

An increasing number of papers have focused on fine-grained object recognition in recent years [5, 6, 1, 7, 8, 9]. In [5], multiple kernel learning is used to combine different types of features and serves as a baseline fine-grained recognition algorithm, and human help is used to discover useful attributes. In [9], a random forest is proposed for fine-grained object recognition that uses different depths of the tree to capture dense spatial information. In [6], a multi-cue combination is used to build discriminative compound words from primitive cues learned independently from training images. In [10], bagging is used to select discriminative ones from the randomly generated templates. In [11], image regions are considered as discriminative attributes and CRF is used to learn the attributes on training set with human in the loop. Pose pooling [12] adapted Poselets [13] to fine-grained recognition problems and learned different poses from fully annotated data. Though deformable parts model [14] is powerful for object detection, it might be insufficient to capture the flexibility and variability in fine-grained tasks considered here [15].

## 3 Unsupervised Learning of Template Model

This section provides an overview of our fine-grained object recognition approach. We discuss the framework of our template based object recognition, describe our template model, and propose an alternative algorithm for learning model parameters.

### 3.1 Template-Based Fine-Grained Object Recognition

Over the last decades, computer vision researchers have done a lot of work in designing effective and efficient patch-level features for object recognition [16, 17, 18, 19, 20, 21, 22]. SIFT is one

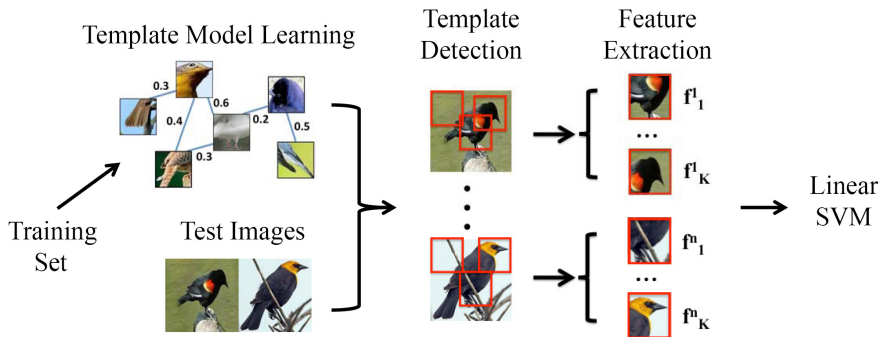

Figure 2: The framework for fine-grained recognition: the recognition pipeline goes from left to right. In the training stage, a template model is learned from training images using Algorithm 1. In the recognition stage, the learned templates are applied to each test image, resulting in aligned image regions. Then image-level feature vectors are extracted as the concatenation of features of all aligned regions. Finally, a linear SVM is used for recognition.

of the most successful features, allowing an image or object to be represented as a bag of SIFT features [16]. However, the ability of such methods is somewhat limited. Patch-level features are descriptive only within spatial context constraints. For example, a cross shape can be a symbol for the Red Cross, Christian religion, or Swiss Army products, depending on the larger spatial context of where it is detected. It is hard to interpret the meaning of patch-level features without considering such spatial contexts. This is even more important for a fine-grained recognition task since common features can be shared by instances from both the same and different object classes. Spatial pyramid models [2, 20, 23] align sub-images/parts that are spatially close by partitioning the whole images into multi-level spatial cells. However the alignments produced by the spatial pyramid are not necessarily correct, since no displacements are allowed in the model (Figure 1).

Here, we use a template model to find correctly-aligned regions from different images, so that comparisons between them are more meaningful. A template represents one type of common shape pattern of an object part, while an object part can be represented by several different templates. Certain shape patterns of two object parts (for instance, a head facing the left and a tail pointing to the right) can frequently be observed in the same image. Our template model is designed to capture both properties of templates and their relationships among templates. Model parameters are learned from a collection of unlabeled images in an unsupervised manner. See sections 3.2 and 3.3 for more details.

Once the templates and their relationship are learned, the fine-grained differences can be aligned based on these quantities. The framework of our template based fine-grained object recognition is illustrated in Figure 2. In the learning stage, Algorithm 1 is used to find the templates. In the recognition stage (from left to right in Figure 2), aligned image regions are extracted from each image using our template detection algorithm. Color-based, normalized color-based, gradient-based, and LBP-based kernel descriptors followed by EMK [4] are then applied to generate feature representations for each region. The image-level feature is the concatenation of feature representations of all detected regions from the corresponding image. Finally, a linear SVM [24] is used for recognition.

## 3.2  Template Model

We start by defining a template model that represents the common shape patterns of object parts and their relationships. A template is an entity that contains features that will match image features for region detection. Let $M = \{\mathbb{T}, \mathcal{W}\}$ be a model that contains a group of templates $\mathbb{T} = \{T_1, T_2, ..., T_K\}$ and their co-occurrence relationships $\mathcal{W} = \{w_{11}\, w_{12}..., w_{KK}\}$, where $K$ is the number of templates, and $w_{ij}$ is between $0$ and $1$. When $w_{ij} = 0$, the two templates $T_i$ and $T_j$ have no co-occurrence relationship.

When a template model is matched to a given image, not all templates within the model are necessarily used. This is because different templates can be associated with the same object part, but

one part only occurs at most once in an image. Our model captures this intuition by making the templates inactive that do not match images very well. To model appearance properties of templates and their relationships, the score function between templates and a given image $I^t$ should capture three aspects: 1) fitness, which computes the similarity of the selected templates and image regions that are most highly matched to them; 2) co-occurrence, which encourages selecting templates that have a high chance of co-occurring in the same image; and 3) diversity, which gives preference to having the selected templates match separated image regions.

**Fitness:** We define a matching score $s^f(T_i, x_i^I)$ to measure the similarity between a template $T_i$ and an image region at location $x_i^I$ in image $I$

$$s^f(T_i, x_i^I) = 1 - \|T_i - R(x_i^I)\|^2 \quad s.t. \ |x_i^I - \overline{x}_i^I| \leq \alpha \tag{1}$$

where $R(x_i^I)$ represents the features of the sub-image in $I$ centered at the location $x_i^I$; $\overline{x}_i^I$ is an initial location associated with the template $T_i$ and $\alpha$ is an upper bound of location variation. Both $x_i^I$ and $\overline{x}_i^I$ are measured by their relative location in image I. If $|x_i^I - \overline{x}_i^I| > \alpha$, the location is too far from the initial location, and the score is set to zero.

The features describing $R(x_i^I)$ should be able to capture common properties of object parts. Since the same type of part from different objects usually share similar shapes, we introduce edge kernel descriptors to capture this common statistic. We first run the Berkeley edge detector [25] to compute the edge map of an image, and then treat it as a grayscale image and extract color kernel descriptors [3] over it. Using these descriptors, we compute $s^f(T_i, x_i^I)$; the higher its value, the better is the match.

Summing up the matching score $s^f(T_i, x_i^I)$ for all templates that are used for image $I$, we obtain a fitness term

$$S^f(\mathbb{T}, \mathcal{X}^I, \mathcal{V}^I) = \sum_{i=1}^{K} v_i^I s^f(T_i, x_i^I) \tag{2}$$

where $\mathcal{V}^I = \{v_1^I, ..., v_K^I\}$ represents the selected template subset for image $I$, $v_i^I = 1$ means that the template $T_i$ is used for image $I$, and $\mathcal{X}^I = \{x_1^I, ..., x_K^I\}$ represents the locations of all templates on image $I$. The more templates that are used, the higher the score is.

**Co-occurrence:** With the observation that certain shape patterns of two or more object parts coexist frequently in the same image, it is desired that templates that have a high chance of co-occurring are selected together. For a given image, the co-occurrence term is used to encourage selecting two templates together, which have a large relation parameter $w_{ij}$. Meanwhile, a $L_1$ penalty term is used to ensure sparsity of the template relation.

$$S^c(\mathcal{W}, \mathcal{V}^I) = \sum_{i=1}^{K}\sum_{j=1}^{K} v_i^I v_j^I w_{ij} - \lambda \sum_{i=1}^{K}\sum_{j=1}^{K} |w_{ij}| \quad s.t. \ 0 \leq w_{ij} \leq 1 \tag{3}$$

**Diversity:** This term is used to enforce spatial relationship constraints on the locations of selected templates. In particular, their locations should not be too close to each other, because we want the learned templates to be diverse, so that they can cover a large range of image shape patterns. So this term sums up a location penalty on the templates,

$$S^d(\mathcal{X}^I, \mathcal{V}^I) = -\sum_{i=1}^{K}\sum_{j=1}^{K} v_i^I v_j^I d(x_i^I, x_j^I) \tag{4}$$

where $d(x_i^I, x_j^I)$ is the location penalty function. We have $d(x_i^I, x_j^I) = \infty$ if $|x_i^I - x_j^I| < \beta$ and $d(x_i^I, x_j^I) = 0$, otherwise. $\beta$ is a distance parameter.

Summing up all three terms defined above: fitness, co-occurrence and diversity terms for all images in the image set $D$, we have the overall score function between templates and images

$$S(\mathbb{T}, \mathcal{W}, \mathcal{X}, \mathcal{V}, D) = \sum_{I \in D} (S^f(\mathbb{T}, \mathcal{X}^I, \mathcal{V}^I) + S^c(\mathcal{W}, \mathcal{V}^I) + S^d(\mathcal{X}^I, \mathcal{V}^I)) \tag{5}$$

where $\mathcal{V} = \{\mathcal{V}^1, \mathcal{V}^2, ..., \mathcal{V}^{|D|}\}$ are template indicators, $\mathcal{X} = \{\mathcal{X}^1, \mathcal{X}^2, ..., \mathcal{X}^{|D|}\}$ are template locations, and $|D|$ is the number of images in the set $D$. The templates and their relations are learned by maximizing the score function $S(\mathbb{T}, \mathcal{W}, \mathcal{X}, \mathcal{V}, D)$ on an image collection $D$.

---

**Algorithm 1** Template Model Learning

---

**input** Image set $D$, maximum iteration $maxiter$, threshold $\epsilon$
**output** Template model $M = \{\mathbb{T}, \mathcal{W}\}$.
  Initialize $\{T_1, T_2, ..., T_K\}$ with training data; initialize $w_{ij} = 0$; $iter = 0$
  **for** $iter < maxiter$ **do**
    update $\mathcal{X}^I, \mathcal{V}^I$ for all $I \in D$ based on equation (6)
    update $\mathbb{T}$ by: $T_i = \sum_{I \in D} v_i^I R(x_i^I) / \sum_{I \in D} v_i^I$ (as in (8))
    update $\mathcal{W}$ to optimize (9)
    **if** $\sum_i |\Delta T_i| < \epsilon$ **then**
      break
    **end if**
    $iter \leftarrow iter + 1$
  **end for**

---

## 3.3 Template Learning

We use an alternating algorithm to optimize (5). The proposed algorithm iterates among three steps:

- updating $\mathcal{X}, \mathcal{V}$ (template detection),
- updating $\mathbb{T}$ (template feature learning), and
- updating $\mathcal{W}$ (template relation learning).

**Template detection:** Given a template model $\{\mathbb{T}, \mathcal{W}\}$, the goal of template detection is to find the template subset $\mathcal{V}$ and their locations $\mathcal{X}$ for all images to maximize equation (5). The second term in $S^c$ in equation (3) is a constant given $\mathcal{W}$. So maximizing (5) is reduced to maximizing the following term for each image $I$ respectively:

$$\max_{\mathcal{X}^I, \mathcal{V}^I} \sum_{i=1}^{K} v_i^I s^f(T_i, x_i^I) + \sum_{i=1}^{K} \sum_{j=1}^{K} v_i^I v_j^I (w_{ij} - d(x_i^I, x_j^I)) \tag{6}$$

The above optimization problem is NP-hard, so a greedy approach is used: the algorithm starts with an empty set, first calculates the scores for all templates, and then selects the template with the largest score. Fixing the locations of all previously selected templates, the next template and its location can be chosen in a similar manner. The procedure is repeated until the object function (6) no longer increases.

**Template feature learning:** The goal of template feature learning is to optimize the templates $\mathbb{T}$ given the relation parameters $\mathcal{W}$ and current template detection results $\mathcal{V}, \mathcal{X}$. When maximizing (5), $S^d$ and $S^c$ are all constants given $\mathcal{V}, \mathcal{X}$ and $\mathcal{W}$. The optimal template $T_i$ can be found by maximizing

$$\max_{T_i} \sum_{I \in D} v_i^I (1 - \|T_i - R(x_i^I)\|^2) \tag{7}$$

which can be solved by the closed form equation

$$T_i = \sum_{I \in D} v_i^I R(x_i^I) / \sum_{I \in D} v_i^I \tag{8}$$

Eq (8) means that the template $T_i$ is updated by the average of the features of all sub-images in $D$ that are detected by the $i$-th template.

**Template relation learning:** The goal here is to assign values to the relation parameters $\mathcal{W}$ given all other parameters ($\mathbb{T}, \mathcal{V}$ and $\mathcal{X}$) for the purpose of maximizing equation (5). Since only $\mathcal{W}$ are optimization parameters, $S^f$ and $S^d$ are both constants. Optimizing (5) is simplified as maximizing

$$\max_{\mathcal{W}} \sum_{i=1}^{K} \sum_{j=1}^{K} w_{ij} \sum_{I \in D} v_i^I v_j^I - \lambda|D| \sum_{i=1}^{K} \sum_{j=1}^{K} |w_{ij}| \tag{9}$$

A $L_1$ regularization solver [26] is used for optimizing this formula.

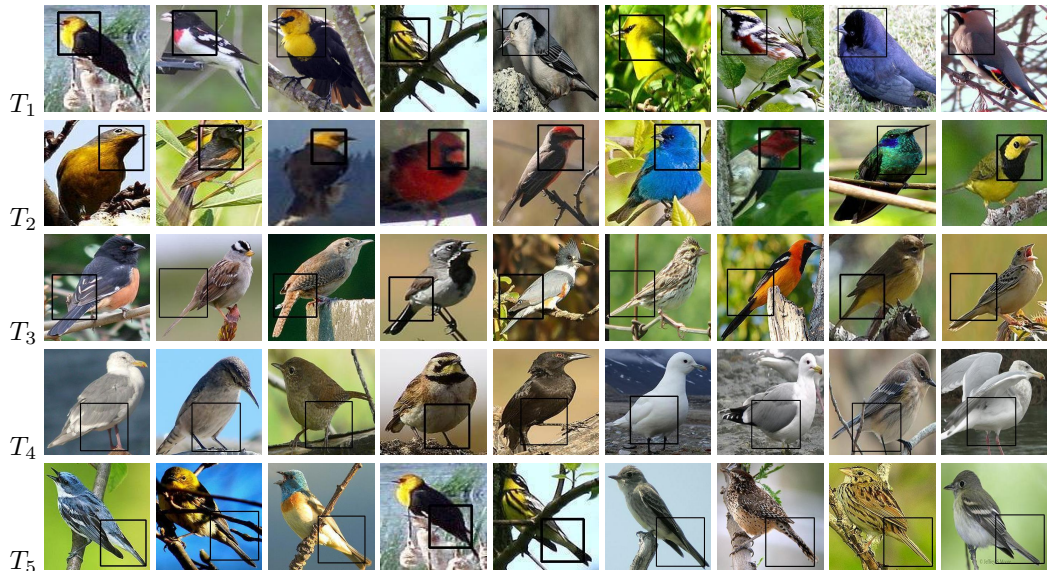

Figure 3: Object parts (black squares) detected by learned templates. Each line shows the parts found by one learned template. The sub-image within the black square has the highest matching score for a given image. Meaningful parts are successfully detected such as heads, backs and tails.

The whole learning procedure is summarized in Algorithm 1. The algorithm starts by initiating $K$ templates with various sizes and initial locations that are are evenly spaced in an image. In each iteration, template detection, template feature learning, and template relation learning are alternated. The iteration continues until the total change of template $\{T_i\}_{i=1}^{K}$ is smaller than a threshold $\epsilon$.

## 4 Experiments

We tested our model on two publicly available datasets: Caltech-UCSD Bird-200 and Stanford Dog. These two datasets are the standard benchmarks to evaluate fine-grained object recognition algorithms. Our experiments suggest that the proposed template model is able to detect the meaningful parts and outperforms the previous work in terms of accuracy.

### 4.1 Features and Settings

We use kernel descriptors (KDES) to capture low-level image statistics: color, shape and texture [3]. In particular, we use four types of kernel descriptors: color-based, normalized color-based, gradient-based, and local-binary-pattern-based descriptors[1]. Color and normalized color kernel descriptors are extracted over RGB images, and gradient and shape kernel descriptors are extracted over gray scale images transformed from the original RGB images. Following the standard parameter setting, we compute kernel descriptors on $16 \times 16$ image patches over dense regular grids with spacing of 8 pixels. For template relation learning, we use a publicly available $L_1$ regularization solver [2]. All images are resized to be no larger than $300 \times 300$ with the height/width ratio preserved.

To learn the template model, we use 34 templates with different sizes. The template size is measured by its ratio to the original image size, such as $1/2$ or $1/3$. Our model has 9 templates with size $1/2$ and 25 with size $1/3$. The initial locations of templates with each template size are evenly spaced grid points in an image. We observe that the learning algorithm converges very fast and usually becomes stable around $15 \sim 20$ iterations.The sparsity level parameter $\lambda$ is set to be $0.1$. Other model parameters are $\alpha = 24$ and $\beta = 32$ pixels. These parameters are optimized by performing cross validation on training set of the Bird dataset. The same parameter setting is then applied to the

Table 1: The table in the left show the classification accuracies (%) obtained by templates with different sizes and numbers on a subset of a full dataset. The accuracy is improved with an increasing template number at the beginning, and become saturated when enough templates are used. With the best template number choices, the combination of templates with different sizes are tested. The table in the right shows the accuracies (%) achieved by different combinations on the full dataset. The combination of 9 templates with size $1/2$ and 25 templates with size $1/3$ performs best (selected using the training set).

| Acc | 1 | 4 | 9 | 16 | 25 | 36 |
|---|---|---|---|---|---|---|
| $T^1$ | **46.1** | 46.1 | 46.1 | 46.1 | 46.1 | 46.1 |
| $T^{\frac{1}{2}}$ | 39.6 | 46.8 | **50.7** | 50.7 | 48.9 | 47.5 |
| $T^{\frac{1}{3}}$ | 33.2 | 42.9 | 41.8 | 43.9 | **44.3** | 44.3 |
| $T^{\frac{1}{4}}$ | 32.1 | 37.5 | **40.4** | 40 | **40.4** | 40 |

| Combination | Acc |
|---|---|
| $9T^{\frac{1}{2}}$ | 27.1 |
| $T^1 + 9T^{\frac{1}{2}}$ | 27.4 |
| $9T^{\frac{1}{2}} + 25T^{\frac{1}{3}}$ | 28.2 |
| $T^1 + 9T^{\frac{1}{2}} + 25T^{\frac{1}{3}} + 25T^{\frac{1}{4}}$ | 28.2 |

Table 2: Effect of sparsity parameter $\lambda$: that the best accuracy is achieved when $\lambda = 0.1$.

| $\lambda$ | 0 | 0.001 | 0.005 | 0.01 | 0.05 | 0.1 | 0.5 | 1 |
|---|---|---|---|---|---|---|---|---|
| Accur | 48.57 | 48.93 | 49.28 | 49.29 | 49.64 | 50.7 | 50 | 48.57 |

Dog dataset. On each region detected by templates, we compute template-level features using EMK features [4]. After obtaining these template-level features, we train a linear support vector machine for fine-grained object recognition.

Notice that there is a slight difference between template detection in the learning phase and in the recognition phase. In the learning phase, only a subset of templates are detected for each image. This is because not all templates can be observed in all images, and each image usually contains only a subset of all possible templates. But in the recognition phase, all templates are selected for detection in order to avoid missing features.

## 4.2  Bird Recognition

Caltech-UCSD Bird-200 [8] is a commonly used dataset for evaluating fine-grained object recognition algorithms. The dataset contains 6033 images from 200 bird species in North America. In each image, the bounding box of a bird is given. Following the standard setting [5], 15 images from each species are used for training and the rest for testing.

**Template learning:** Figure 3 visualizes the rectangles/parts detected by the learned templates. The feature in each template consists of a vector of real numbers. As can be seen, the learned templates successfully find the meaningful parts of birds, though the appearances of these parts are very different. For examples, the head parts detected by $T_1$ have quite different colors and textures, suggesting the robustness of the proposed template model.

**Sparsity parameter $\lambda$:** We tested different values for the sparsity level parameter $\lambda$ on a subset of 20 categories (from the training set) for efficiency. If $\lambda = 0$, there is no penalty on the relation parameters $\mathcal{W}$, thus all weights $w_{ij}$ are set to 1 when the template model is learned. If $\lambda \geq 1$, the penalty on the relation parameters is large enough that all $w_{ij}$ are set to 0 after learning. In both these cases, the template models are equivalent to a simplified model without the co-occurrence term in (3). If $\lambda$ is a number between 0 and 1, test results in Table 2 show that the best accuracy is achieved when $\lambda = 0.1$.

**Template size and number choices:** We tested the effect of the number and size of the templates on the recognition accuracy. All the results are obtained on a subset of 20 categories for efficiency. When the template size is 1, the accuracy is the same with an arbitrary template number, because template detection will return the same results. For templates whose size is smaller than 1, the results obtained with different numbers of templates are shown in Table 1 *left*. Based on these results, we selected a template number for each template size for further experiments: one template with size 1, 9 templates with size $1/2$, 25 templates with size $1/3$, and 25 templates with size $1/4$. The results obtained by the combinations of templates with different sizes (each with its optimal template number) on the full dataset are shown in Table 1 *right*. The highest accuracy is achieved by the

Table 3: Comparisons on Caltech-UCSD Bird-200. Our template model is compared to the recently proposed fine-grained recognition algorithms. The performance is measured in terms of accuracy.

| MKL [5] | LLC [9] | Rand-forest [9] | Multi-cue [6] | KDES [3] | This work |
|---------|---------|-----------------|---------------|----------|-----------|
| 19.0 | 18.0 | 19.2 | 22.4 | 26.4 | **28.2** |

Table 4: Comparisons on Stanford Dog Dataset. Our approach is compared to a baseline algorithm in [27] and KDES with spatial pyramid. We give the results of the proposed template model with two types of templates: edge templates and texture templates.

| Methods | SIFT [27] | KDES [3] | Edge templates | Texture templates |
|---------|-----------|----------|----------------|-------------------|
| Accuracy(%) | 22.0 | 36.0 | **38.0** | 36.9 |

combination of 9 templates with size $1/2$ and 25 templates with size $1/3$. Our further experiments suggest that adding more templates only slightly improves the recognition accuracy.

**Running time:** Our algorithm is efficient. With a non-optimized version of the algorithm, in the training stage, each iteration takes $2 \sim 3$ minutes to update. In the test stage, it takes $3 \sim 5$ seconds to process each image, including template detection, feature extraction and classification. This is fast enough for an on-line recognition task.

**Comparisons with the state-of-the-art algorithms:** We compared our model with four recently published algorithms for fine-grained object recognition: multiple kernel learning [5], random forest [9], LLC [9], and multi-cue [6] in Table 3. We also compared our model to KDES [3] with spatial pyramid, a strong baseline in terms of accuracy.

We observe that KDES with spatial pyramid works well on this dataset, and the proposed template model works even better. The template model achieves 28.2% accuracy, about 6 percents higher than the best results reported in the previous work and about 2 percents higher than KDES with spatial pyramid. This accuracy is comparable with the recently proposed pose pooling approach [12] where labeled parts are used to train and test models; this is not required for our template model.

## 4.3 Dog Recognition

The Stanford Dogs dataset is another benchmark dataset for fine-grained image categorization recently introduced in [27]. The dataset contains $20,580$ images of 120 breeds of dogs from around the world. Bounding boxes of dogs are provided for all images in the dataset. This dataset is a good complement to the Caltech-UCSD Bird200 due to more images in each category: around 200 images per class versus 30 images per class in Bird200. Following the standard setting [27], 100 images from each category are used for training and the rest for testing.

**Comparisons with the state-of-art algorithms:** We compared our model with a baseline algorithm [27] and KDES with spatial pyramid on this dataset. For the dog datasets, we also tried using the local binary pattern KDES to learn templates instead of the edge KDES due to the relative consistent textures in dog images. Our experiments show that the template learning with the edge KDES works better than that with the local binary pattern KDES, suggesting that the edge information is a stable cue to learn templates. Notice that the accuracy achieved by our template model is 16 percent higher than the best published results so far.

## 5 Conclusion

We have proposed a template model for fine-grained object recognition. The template model learns a group of templates by jointly considering fitness, co-occurrence and diversity between the templates and images, and the learned templates are used to align image regions that contain the same object parts. Our experiments show that the proposed template model has achieved higher accuracy than the state-of-the-art fine-grained object recognition algorithms on the two standard benchmarks: Caltech-UCSD Bird-200 and Standford Dogs. In the future, we plan to learn the features that are suitable for detecting object parts and incorporate the geometric information into the template relationships.

## Footnotes

[1] http://www.cs.washington.edu/ai/Mobile_Robotics/projects/kdes/

[2] http://www.di.ens.fr/~mschmidt/Software/L1General.html

# References

[1] Farrell, R., Oza, O., Zhang, N., Morariu, V., Darrell, T., Davis, L.: Birdlets: subordinate categorization using volumetric primitives and pose-normalized appearance. ICCV (2011)

[2] Lazebnik, S., Schmid, C., Ponce, J.: Beyond bags of features: Spatial pyramid matching for recognizing natural scene categories. CVPR (2006)

[3] Bo, L., Ren, X., Fox, D.: Kernel Descriptors for Visual Recognition. NIPS (2010)

[4] Bo, L., Sminchisescu, C.: Efficient match kernel between sets of features for visual recognition. NIPS (2009)

[5] Branson, S., Wah, C., Babenko, B., Schroff, F., Welinder, P., Perona, P., Belongie, S.: Visual recognition with humans in the loop. ECCV (2010)

[6] Khan, F., van de Weijer, J., Bagdanov, A., Vanrell, M.: Portmanteau vocabularies for multi-cue image representations. NIPS (2011)

[7] Wah, C., Branson, S., Perona, P., Belongie, S.: Interactive localization and recognition of fine-grained visual categories. ICCV (2011)

[8] Welinder, P., Branson, S., Mita, T., Wah, C., Schroff, F., Belongie, S., Perona, P.: Caltech-ucsd birds 200. Technical Report CNS-TR-201, Caltech (2010)

[9] Yao, B., Khosla, A., Fei-Fei, L.: Combining randomization and discrimination for fine-grained image categorization. CVPR (2011)

[10] Yao, B., Bradski, G., Fei-Fei, L.: A codebook-free and annotation-free approach for fine-grained image categorization. CVPR (2012)

[11] Duan, K., Parikh, D., Crandall, D., Grauman, K.: Discovering localized attributes for fine-grained recognition. CVPR (2012)

[12] Zhang, N., Farrell, R., Darrell, T.: Pose pooling kernels for sub-category recognition. CVPR (2012)

[13] Bourdev, L., Malik, J.: Poselets: body partddetectors trained using 3d human pose annotations. ICCV (2009)

[14] Felzenszwalb, P., Girshick, R., McAllester, D., Ramanan, D.: Object detection with discriminatively trained part based models. IEEE Transactions on Pattern Analysis and Machine Intelligence **32** (2010)

[15] Parkhi, O., Vedaldi, A., Zisserman, A., Jawahar, C.: Cats and dogs. CVPR (2012)

[16] Lowe, D.: Distinctive image features from scale-invariant keypoints. IJCV **60** (2004)

[17] Lee, H., Battle, A., Raina, R., Ng, A.: Efficient sparse coding algorithms. NIPS (2007)

[18] Yang, J., Yu, K., Gong, Y., Huang, T.: Linear spatial pyramid matching using sparse coding for image classification. CVPR (2009)

[19] Wang, J., Yang, J., Yu, K., Lv, F., Huang, T., Guo, Y.: Locality-constrained linear coding for image classification. CVPR (2010)

[20] Boureau, Y., Bach, F., LeCun, Y., Ponce, J.: Learning mid-level features for recognition. CVPR (2010)

[21] Coates, A., Ng, A.: The importance of encoding versus training with sparse coding and vector quantization. ICML (2011)

[22] Yu, K., Lin, Y., Lafferty, J.: Learning image representations from the pixel level via hierarchical sparse coding. CVPR (2011)

[23] Boureau, Y., Ponce, J.: A theoretical analysis of feature pooling in visual recognition. ICML (2010)

[24] Chang, C., Lin, C.: LIBSVM: a library for support vector machines. (2001)

[25] Maire, M., Arbelaez, P., Fowlkes, C., Malik, J.: Using contours to detect and localize junctions in natural images. CVPR (2008)

[26] Schmidt, M., Fung, G., Rosales, R.: Optimization methods for $L_1$-regularization. UBC Technical Report (2009)

[27] Khosla, A., Jayadevaprakash, N., Yao, B., Fei-Fei, L.: Novel dataset for fine-grained image categorization. First Workshop on Fine-Grained Visual Categorization, CVPR (2011)

